# A neural network implementing optimal state estimation based on dynamic spike train decoding

**Omer Bobrowski**[1]**, Ron Meir**[1]**, Shy Shoham**[2] **and Yonina C. Eldar**[1]
Department of Electrical Engineering[1] and Biomedical Engineering[2]
Technion, Haifa 32000, Israel
{bober@tx},{rmeir@ee},{sshoham@bm},{yonina@ee}.technion.ac.il

## Abstract

It is becoming increasingly evident that organisms acting in uncertain dynamical environments often employ exact or approximate Bayesian statistical calculations in order to continuously estimate the environmental state, integrate information from multiple sensory modalities, form predictions and choose actions. What is less clear is how these putative computations are implemented by cortical neural networks. An additional level of complexity is introduced because these networks observe the world through spike trains received from primary sensory afferents, rather than directly. A recent line of research has described mechanisms by which such computations can be implemented using a network of neurons whose activity directly represents a probability distribution across the possible "world states". Much of this work, however, uses various approximations, which severely restrict the domain of applicability of these implementations. Here we make use of rigorous mathematical results from the theory of continuous time point process filtering, and show how optimal real-time state estimation and prediction may be implemented in a general setting using linear neural networks. We demonstrate the applicability of the approach with several examples, and relate the required network properties to the statistical nature of the environment, thereby quantifying the compatibility of a given network with its environment.

## 1 Introduction

A key requirement of biological or artificial agents acting in a random dynamical environment is estimating the state of the environment based on noisy observations. While it is becoming clear that organisms employ some form of Bayesian inference, it is not yet clear how the required computations may be implemented in networks of biological neurons. We consider the problem of a system, receiving multiple state-dependent observations (possibly arising from different sensory modalities) in the form of spike trains, and construct a neural network which, based on these noisy observations, is able to optimally estimate the probability distribution of the hidden world state.

The present work continues a line of research attempting to provide a probabilistic Bayesian framework for optimal dynamical state estimation by biological neural networks. In this framework, first formulated by Rao (e.g., [8, 9]), the time-varying probability distributions are represented in the neurons' activity patterns, while the network's connectivity structure and intrinsic dynamics are responsible for performing the required computation. Rao's networks use linear dynamics and discrete time to approximately compute the log-posterior distributions from noisy continuous inputs (rather than actual spike trains). More recently, Beck and Pouget [1] introduced networks in which the neurons directly represent and compute the posterior probabilities (rather than their logarithms) from discrete-time approximate firing rate inputs, using non-linear mechanisms such as multiplicative interactions and divisive normalization. Another relevant line of work, is that of Brown and colleagues as well as others (e.g., [4, 11, 13]) where approximations of optimal dynamical estima-

tors from spike-train based inputs are calculated, however, without addressing the question of neural implementation.

Our approach is formulated within a continuous time point process framework, circumventing many of the difficulties encountered in previous work based on discrete time approximations and input smoothing. Moreover, using tools from the theory of continuous time point process filtering (e.g., [3]), we are able to show that a linear system suffices to yield the *exact* posterior distribution for the state. The key element in the approach is switching from posterior distributions to a new set of functions which are simply non-normalized forms of the posterior distribution. While posterior distributions generally obey non-linear differential equations, these non-normalized functions obey a linear set of equations, known as the Zakai equations [15]. Intriguingly, these linear equations contain the full information required to reconstruct the optimal posterior distribution! The linearity of the exact solution provides many advantages of interpretation and analysis, not least of which is an exact solution, which illustrates the clear distinction between observation-dependent and independent contributions. Such a separation leads to a characterization of the system performance in terms of prior knowledge and real-time observations. Since the input observations appear directly as spike trains, no temporal information is lost. The present formulation allows us to consider inputs arising from several sensory modalities, and to determine the contribution of each modality to the posterior estimate, thereby extending to the temporal domain previous work on optimal multi-modal integration, which was mostly restricted to the static case. Inherent differences between the modalities, related to temporal delays and different shapes of tuning curves can be incorporated and quantified within the formalism.

In a historical context we note that a mathematically rigorous approach to point process based filtering was developed during the early 1970s following the seminal work of Wonham [14] for finite state Markov processes observed in Gaussian noise, and of Kushner [7] and Zakai [15] for diffusion processes. One of the first papers presenting a mathematically rigorous approach to nonlinear filtering in continuous time based on point process observations was [12], where the exact nonlinear differential equations for the posterior distributions are derived. The presentation in Section 4 summarizes the main mathematical results initiated by the latter line of research, adapted mainly from [3], and serves as a convenient starting point for many possible extensions.

## 2    A neural network as an optimal filter

Consider a dynamic environment characterized at time $t$ by a state $X_t$, belonging to a set of $N$ states, namely $X_t \in \{s_1, s_2, \ldots, s_N\}$. We assume the state dynamics is Markovian with generator matrix $Q$. The matrix $Q$, $[Q]_{ij} = q_{ij}$, is defined [5] by requiring that for small values of $h$, $\Pr[X_{t+h} = s_i | X_t = s_i] = 1 + q_{ii}h + o(h)$ and $\Pr[X_{t+h} = s_j | X_t = s_i] = q_{ij}h + o(h)$ for $j \neq i$. The normalization requirement is that $\sum_j q_{ij} = 0$. This matrix controls the process' infinitesimal progress according to $\dot{\boldsymbol{\pi}}(t) = \boldsymbol{\pi}(t)Q$, where $\pi_i(t) = \Pr[X_t = s_i]$.

The state $X_t$ is not directly observable, but is only sensed through a set of $M$ random state-dependent observation point processes $\{N_t^{(k)}\}_{k=1}^M$. We take each point process $N_t^{(k)}$ to represent the spiking activity of the $k$-th sensory cell, and assume these processes to be doubly stochastic Poisson counting processes[1] with state-dependent rates $\lambda_k(X_t)$. These processes are assumed to be independent, *given* the current state $X_t$. The objective of state estimation (a.k.a. nonlinear filtering) is to obtain a differential equation for the posterior probabilities

$$p_i(t) \triangleq \Pr\left[X_t = s_i \,\middle|\, N_{[0,t]}^{(1)}, \ldots, N_{[0,t]}^{(M)}\right], \tag{1}$$

where $N_{[0,t]}^{(k)} = \{N_s^{(k)}\}_{0 \leq s \leq t}$. In the sequel we denote $Y_0^t \triangleq \left\{N_{[0,t]}^{(1)}, \ldots, N_{[0,t]}^{(M)}\right\}$, and refer the reader to Section 4 for precise mathematical definitions.

We interpret the rate $\lambda_k$ as providing the tuning curve for the $k$-th sensory input. In other words, the $k$-th sensory cell responds with strength $\lambda_k(s_i)$ when the input state is $X_t = s_i$. The required differential equations for $p_i(t)$ are considerably simplified, with no loss of information [3], by considering a set of non-normalized 'probability functions' $\rho_i(t)$, such that $p_i(t) = \rho_i(t) / \sum_{j=1}^N \rho_j(t)$.

Based on the theory presented in Section 4 we obtain

$$\dot{\rho}_i(t) = \sum_{j=1}^{N} Q_{ji}\rho_j(t) + \sum_{k=1}^{M} (\lambda_k(s_i) - 1) \left[\sum_n \delta(t - t_n^k) - 1\right] \rho_i(t), \qquad (2)$$

where $\{t_n^k\}$ denote the spiking times of the $k$-th sensory cell. This equation can be written in vector form by defining

$$\Lambda_k = \text{diag}(\lambda_k(s_1) - 1, \lambda_k(s_2) - 1, \ldots \lambda_k(s_N) - 1) \qquad ; \qquad \Lambda = \sum_{k=1}^{M} \Lambda_k, \qquad (3)$$

and $\boldsymbol{\rho} = (\rho_1, \ldots, \rho_N)$, leading to

$$\dot{\boldsymbol{\rho}}(t) = (Q - \Lambda)^{\top}\boldsymbol{\rho}(t) + \sum_{k=1}^{M} \Lambda_k \sum_n \delta(t - t_n^k)\boldsymbol{\rho}(t). \qquad (4)$$

Equations (2) and (4) can be interpreted as the activity of a linear neural network, where $\rho_i(t)$ represents the firing rate of neuron $i$ at time $t$, and the matrix $(Q - \Lambda)^{\top}$ represents the synaptic weights (including self-weights); see Figure 1 for a graphical display of the network. Assuming that the tuning functions $\lambda_k$ are unimodal, decreasing in all directions from some maximal value (e.g., Gaussian or truncated cosine functions), we observe from (2) that the impact of an input spike at time $t$ is strongest on cell $i$ for which $\lambda_k(s_i)$ is maximal, and decreases significantly for cells $j$ for which $s_j$ is 'far' from $s_i$. This effect can be modelled using excitatory/inhibitory connections, where neurons representing similar states excite each other, while neurons corresponding to very different states inhibit each other (e.g., [2]). This issue will be elaborated on in future work.

Several observations are in place regarding (4). (i) The solution of (4) provides the optimal posterior state estimator given the spike train observations, i.e., no approximation is involved. (ii) The equations are linear even though the equations obeyed by the posterior probabilities $p_i(t)$ are nonlinear. (iii) The temporal evolution breaks up neatly into an observation-independent term, which can be conceived of as implementing a Bayesian dynamic prior, and an observation-dependent term, which contributes each time a spike occurs. Note that a similar structure was observed recently in [1]. (iv) The observation process affects the posterior estimate through two terms. First, input processes with strong spiking activity, affect the activity more strongly. Second, the $k$-th input affects most strongly the components of $\boldsymbol{\rho}(t)$ corresponding to states with large values of the tuning curve $\lambda_k(s_i)$. (v) At this point we assume that the matrix $Q$ is known. In a more general setting, one can expect $Q$ to be *learned* on a slower time scale, through interaction with the environment. We leave this as a topic for future work.

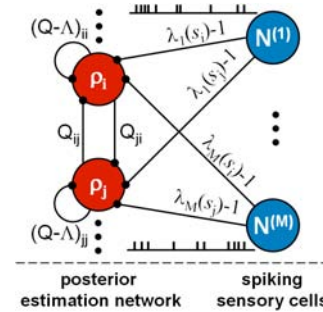

Figure 1: A graphical depiction of the network implementing optimal filtering of $M$ spike train inputs.

**Multi-modal inputs** A multi-modal scenario may be envisaged as one in which a subset of the sensory inputs arises from one modality (e.g., visual) while the remaining inputs arise from a different sensory modality (e.g., auditory). These modalities may differ in the shapes of their receptive fields, their response latencies, etc. The framework developed above is sufficiently general to deal with any number of modalities, but consider for simplicity just two modalities, denoted by $V$ and $A$. It is straightforward to extend the derivation of (4), leading to

$$\dot{\boldsymbol{\rho}}(t) = (Q - \Lambda^v - \Lambda^a)^{\top}\boldsymbol{\rho}(t) + \left\{\sum_{k=1}^{M_v} \Lambda_k^v \sum_n \delta(t - t_n^{v,k}) + \sum_{k=1}^{M_a} \Lambda_k^a \sum_n \delta(t - t_n^{a,k})\right\}\boldsymbol{\rho}(t). \qquad (5)$$

**Prediction** The framework can easily be extended to prediction, defined as the problem of calculating the future posterior distribution $p_i^h(t) = \Pr[X_{t+h} = s_i|Y_0^t]$. It is easy to show that the

non-normalized probabilities $\boldsymbol{\rho}^h(t)$ can be calculated using the vector differential equation

$$\dot{\boldsymbol{\rho}}^h(t) = (Q - \tilde{\Lambda})^\top \boldsymbol{\rho}^h(t) + \sum_{k=1}^M \tilde{\Lambda}_k \sum_n \delta(t - t_n^k)\boldsymbol{\rho}^h(t), \tag{6}$$

with the initial condition $\boldsymbol{\rho}^h(0) = e^{hQ^\top}\boldsymbol{\rho}(0)$, and where $\tilde{\Lambda}_k = e^{hQ^\top}\Lambda_k e^{-hQ^\top}$. Interestingly, the equations obtained are identical to (4), except that the system parameters are modified.

**Simplified equation** When the tuning curves of the sensory cells are uniformly distributed Gaussians (e.g., spatial receptive fields), namely $\lambda_k(x) = \lambda_{\max}\exp(-(x - k\Delta x)^2/2\sigma^2)$, it can be shown [13] that for small enough $\Delta x$, and a large number of sensory cells, $\sum_{k=1}^M \lambda_k(x) \approx \beta$ for all $x$, implying that $\Lambda = \sum_k \Lambda_k \approx (\beta - M)\mathbf{I}$. Therefore the matrix $\Lambda$ has no effect on the solution of (4), except for an exponential attenuation that is applied to all the cells simultaneously. Therefore, in cases where the number of sensory cells is large, $\Lambda$ can be omitted from (4). This means that between spike arrivals, the system behaves solely according to the a-priori knowledge about the world, and when a spike arrives, this information is reshaped according to the firing cell's tuning curve.

## 3 Theoretical Implications and Applications

Viewing (4) we note that between spike arrivals, the input has no effect on the system. Therefore, the inter-arrival dynamics is simply $\dot{\boldsymbol{\rho}}(t) = (Q - \Lambda)^\top \boldsymbol{\rho}(t)$. Defining $t_n$ as the $n$-th arrival time of a spike from any one of the sensors, the solution in the interval $(t_n, t_{n+1})$ is

$$\boldsymbol{\rho}(t) = e^{(t-t_n)(Q-\Lambda)^\top}\boldsymbol{\rho}(t_n).$$

When a new spike arrives from the $k$-th sensory neuron at time $t_n$ the system is modified within an infinitesimal window of time as

$$\rho_i(t_n^+) = \rho_i(t_n^-) + \rho_i(t_n^-)(\lambda_k(s_i) - 1) = \rho_i(t_n^-)\lambda_k(s_i). \tag{7}$$

Thus, at the exact time of a spike arrival from the $k$-th sensory cell, the vector $\boldsymbol{\rho}$ is reshaped according to the tuning curve of the input cell that fired this spike. Assuming $n$ spikes occurred before time $t$, we can derive an *explicit* solution to (4), given by

$$\boldsymbol{\rho}(t) = e^{(t-t_n)(Q-\Lambda)^\top}\prod_{i=1}^n (I + \Lambda_{k(t_i)})e^{(t_i - t_{i-1})(Q-\Lambda)^\top}\boldsymbol{\rho}(0), \tag{8}$$

where $k(t_i)$ is the index of the cell that fired at $t_i$, $I$ is the identity matrix, and we assumed initial conditions $\boldsymbol{\rho}(0)$ at $t_0 = 0$.

### 3.1 Demonstrations

We demonstrate the operation of the system on several synthetic examples. First consider a small object moving back and forth on a line, jumping between a set of discrete states, and being observed by a retina with $M$ sensory cells. Each world state $s_i$ describes the particle's position, and each sensory cell $k$ generates a Poisson spike train with rate $\lambda_k(X_t)$, taken to be a Gaussian $\lambda_{\max}\exp\left(-(x - x_k)^2/2\sigma^2\right)$. Figure 2(a) displays the motion of the particle for a specific choice of matrix $Q$, and 2(b) presents the spiking activity of 10 position sensitive sensory cells. Finally, Figure 2(c) demonstrates the tracking ability of the system, where the width of the gray trace corresponds to the prediction confidence. Note that the system is able to distinguish between 25 different states rather well with only 10 sensory cells.

In order to enrich the systems's estimation capabilities, we can include additional parameters within the state of the world. Considering the previous example, we create a larger set of states: $\tilde{s}_{ij} = (s_i, d_j)$, where $d_j$ denotes the current movement direction (in this case $d_1$=up, $d_2$=down). We add a population of sensory cells that respond differently to different movement directions. This lends further robustness to the state estimation. As can be seen in Figure 2(d)-(f), when for some reason the input of the sensory cells is blocked (and the sensory cells fire spontaneously) the system estimates a movement that continues in the same direction. When the blockade is removed, the system is re-synchronized with the input. It can be seen that even during periods where sensory input is absent, the general trend is well predicted, even though the estimated uncertainty is increased.

By expanding the state space it is also possible for the system to track multiple objects simultaneously. In figure 2(g)-(i) we present tracking of two simultaneously moving objects. This is done simply by creating a new state space, $s_{ij} = (s_i^1, s_j^2)$, where $s_i^k$ denotes the state (location) of the $k$-th object.

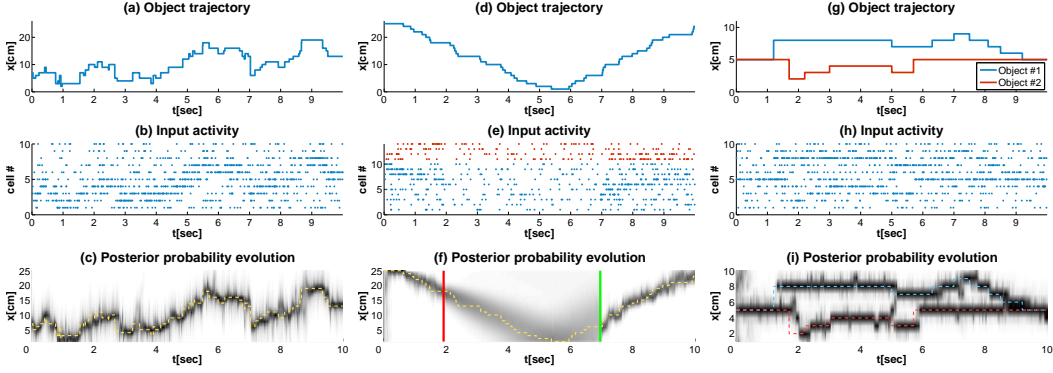

Figure 2: Tracking the motion of an object in 1D. (a) The object's trajectory. (b) Spiking activity of 10 sensory cells. (c) Decoded position estimation with confidence interval. Each of the 10 sensory cells has a Gaussian tuning curve of width $\sigma = 2$ and maximal firing rate $\lambda_{\max} = 25$.(d)-(f) Tracking based on position and direction information. The red bar marks the time when the input was blocked, and the green bar marks the time when the blockade was removed. Here we used 10 place-cells and 4 direction-cells (marked in red). (g)-(i) Tracking of two objets simultaneously. The network activity in (i) represents $\Pr\left[X_t^1 = s_i \vee X_t^2 = s_i | Y_0^t\right]$.

## 3.2   Behavior Characterization

The solution of the filtering equations (4) depends on two processes, namely the recurrent dynamics due to the first term, and the sensory input arising from the second term. Recall that the connectivity matrix $Q$ is essentially the generator matrix of the state transition process, and as such, incorporates prior knowledge about the world dynamics. The second term, consisting of the sensory input, contributes to the state estimator update every time a spike occurs. Thus, a major question relates to the interplay between the a-priori knowledge embedded in the network through $Q$ and the incoming sensory input. In particular, an important question relates to tailoring the system parameters (e.g., the tuning curves $\lambda_k$), to the properties of the external world. As a simple characterization of the generator matrix $Q$, we consider the diagonal and non-diagonal terms. The diagonal term $q_{ii}$ is related to the average time spent in state $i$ through $\mathbf{E}[T_i] = -1/q_{ii}$ [5], and thus we define $\tau(Q) = -\left(q_{11}^{-1} + \cdots + q_{NN}^{-1}\right)/N$, as a measure of the transition frequency of the process, where small values of $\tau$ correspond to a rapidly changing process. A second relevant measure relates to the regularity in the transition between the states. To quantify this consider a state $i$, and define a probability vector $\mathbf{q}_i$ consisting of the $N-1$ elements $\{Q_{ij}\}$, $j \neq i$, normalized so that the sum of the elements is 1. The entropy of $\mathbf{q}_i$ is a measure for the state transition irregularity from state $i$, and we define $H(Q)$ as the average of this entropy over all states. In summary, we lump the main properties of $Q$ into $\tau(Q)$, related to the rapidity of the process, and $H(Q)$, measuring the transition regularity. Clearly, these variables are but one heuristic choice for characterizing the Markov process dynamics, but they will enable us to relate the 'world dynamics' to the system behavior. The sensory input influence on the system is controlled by the tuning curves. To simplify the analysis we assume uniformly placed Gaussian tuning curves, $\lambda_k(x) = \lambda_{\max} \exp\left(-(x - k\Delta x)^2/2\sigma^2\right)$, which can be characterized by two parameters - the maximum value $\lambda_{\max}$ and the width $\sigma$. Note, however that our model does not require any special constraints on the tuning curves.

Figure 3 examines the system performance under different world setups. We measure the performance using the $L_1$ error of the maximum aposteriori (MAP) estimator built from the posterior distribution generated by the system. The MAP estimator is obtained by selecting the cell with the highest firing activity $\rho_i(t)$, is optimal under the present setting (leading to the minimal probability of error), and can be easily implemented in a neural network by a Winner-Take-All circuit. The choice of the $L_1$ error is justified in this case since the states $\{s_i\}$ represent locations on a line,

thereby endowing the state space with a distance measure. In figure 3(a) we can see that as $\tau(Q)$ increases, the error diminishes, an expected result, since slower world dynamics are easier to analyze. The effect of $H(Q)$ is opposite - lower entropy implies higher confidence in the next state. Therefore we can see that the error increases with $H(Q)$ (fig. 3(b)). The last issue we examine relates to the behavior of the system when an incorrect $Q$ matrix is used (i.e., the world model is incorrect). It is clear from figure 3(c) that for low values of $M$ (the number of sensory cells), using the wrong $Q$ matrix increases the error level significantly. However as the value of $M$ increases, the differences are reduced. This phenomenon is expected, since the more observations are available about the world, the less weight need be assigned to the a-priori knowledge.

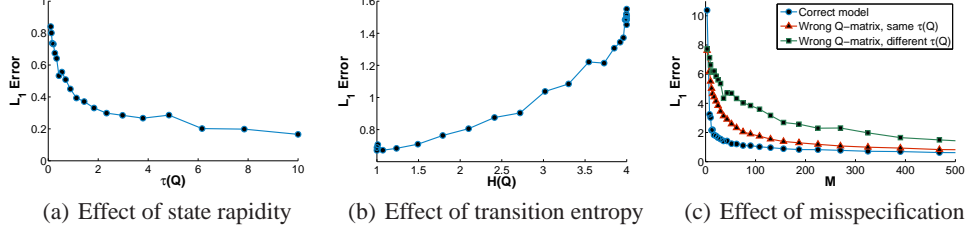

(a) Effect of state rapidity     (b) Effect of transition entropy     (c) Effect of misspecification

Figure 3: State estimation error for different world dynamics and model misspecification. For (a) and (b) $M = 17$, $N = 17$, $\sigma = 3$, $\lambda_{\max} = 50$, and for (c) $N = 25$, $\sigma = 3$, $\lambda_{\max} = 50$.

In figure 4 we examine the effect of the tuning curve parameters on the system's performance. Given a fixed number of input cells, if the tuning curves are too narrow (fig. 4(a) top), they will not cover the entire state space. On the other hand, if the tuning curves are too wide (fig. 4(a) bottom) the cell's response is very similar for all states. Therefore we get an error function that has a local minimum (fig. 4(b)). It remains for future work to determine what is the optimal value of $\sigma$ for a given model. The effect of different values of $\lambda_{\max}$ is obvious - higher values of $\lambda_{\max}$ lead to more spikes per sensory cell which increases the system's accuracy. Clearly, under physiological conditions, where high firing rates are energetically costly, we would expect a tradeoff between accuracy and energy expenditure.

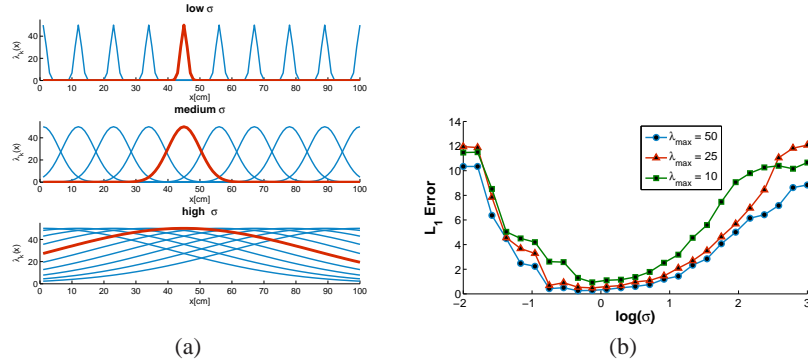

(a)                    (b)

Figure 4: The effect of the tuning curves parameters on performance.

## 4   Mathematical Framework and Derivations

We summarize the main mathematical results related to point process filtering, adapted mainly from [3]. Consider a finite-state continuous-time Markov process $X_t \in \{s_1, s_2, \ldots, s_N\}$ with a generator matrix $Q$ that is being observed via a set of (doubly stochastic) Poisson processes with state-dependent rate functions $\lambda_k(X_t)$, $k = 1, \ldots, M$.

Consider first a single point process observation $N_0^t = \{N_s\}_{0 \leq s \leq t}$. We denote the joint probability law for the state and observation process by $P_1$. The objective is to derive a differential equation for the posterior probabilities (1). This is the classic *nonlinear filtering* problem from systems theory

(e.g. [6]). More generally, the problem can be phrased as computing $\mathbf{E}_1[f(X_t)|N_0^t]$, where, in the case of (1), $f$ is a vector function, with components $f_i(x) = \quad [x = s_i]$.

We outline the derivation required to obtain such an equation, using a method referred to as *change of measure* (e.g., [3]). The basic idea is to replace the computationally hard evaluation of $\mathbf{E}_1[f(X_t)|N_0^t]$, by a tractable computation based on a simple probability law. Consider two probability spaces $(\Omega, \mathcal{F}, \mathcal{P}_1)$ and $(\Omega, \mathcal{F}, \mathcal{P}_\infty)$ that differ only in their probability measures. $P_1$ is said to be *absolutely continuous* with respect to $P_0$ (denoted by $P_1 \ll P_0$), if for all $A \in \mathcal{F}$, $P_0(A) = 0 \Rightarrow P_1(A) = 0$. Assuming $P_1 \ll P_0$, it can be proved that there exists a random variable $L(\omega), \omega \in \Omega$, such that for all $A \in \mathcal{F}$,

$$P_1(A) = \mathbf{E}_0[1_A L] = \int_A L(\omega) dP_0(\omega), \tag{9}$$

where $\mathbf{E}_0$ denotes the expectation with regard to $P_0$. The random variable $L$ is called the *Radon-Nykodim derivative* of $P_1$ with respect to $P_0$, and is denoted by $L(\omega) = dP_1(\omega)/dP_0(\omega)$.

Consider two continuous-time random processes - $X_t, N_t$, that have different interpretation under the different probability measures - $P_0, P_1$:

$$P_0 : \begin{cases} X_t \text{ is a finite-state Markov process}, X_t \in \{s_1, s_2, \ldots, s_N\}. \\ N_t \text{ is a Poisson process with a constant rate of 1, independent of } X_t \end{cases}, \tag{10}$$

$$P_1 : \begin{cases} X_t \text{ is a finite-state Markov process}, X_t \in \{s_1, s_2, \ldots, s_N\}. \\ N_t \text{ is a doubly-stochastic Poisson process with rate function: } \lambda(X_t) \end{cases}. \tag{11}$$

The following avatar of Bayes' formula (eq. 3.5 in chap. 6 of [3]), supplies a way to calculate the conditional expectation $\mathbf{E}_1[f(X_t)|N_0^t]$ based on $P_1$ in terms of an expectation w.r.t. $P_0$,

$$\mathbf{E}_1[f(X_t)|N_0^t] = \frac{\mathbf{E}_0[L_t f(X_t)|N_0^t]}{\mathbf{E}_0[L_t|N_0^t]}, \tag{12}$$

where $L_t = dP_{1,t}/dP_{0,t}$, and $P_{0,t}$ and $P_{1,t}$ are the restrictions of $P_0$ and $P_1$, respectively, to the sigma-algebra generated by $\{N_0^t, X_0^\infty\}$. We refer the reader to [3] for precise definitions.

Using (1) and (12) we have

$$p_i(t) = \mathbf{E}_1[f_i(X_t)|N_0^t] = \frac{\mathbf{E}_0[L_t f_i(X_t)|N_0^t]}{\mathbf{E}_0[L_t|N_0^t]}. \tag{13}$$

Since the denominator is independent of $i$, it can be regarded as a normalization factor. Thus, defining $\rho_i(t) \triangleq \mathbf{E}_0[L_t f_i(X_t)|N_0^t]$, it follows that $p_i(t) = \rho_i(t)/\sum_{j=1}^N \rho_j(t)$.

Based on the above derivation, one can show ([3], chap. 6.4) that $\{\rho_i(t)\}$ obey the stochastic differential equation (SDE)

$$d\rho_i(t) = \sum_{j=1}^N Q_{ji}\rho_j(t)dt + (\lambda(s_i) - 1)\rho_i(t)(dN_t - dt). \tag{14}$$

A SDE of the form $d\rho(t) = a(t)dt + b(t)dN_t$ should be interpreted as follows. If at time $t$, no jump occurred in the counting process $N_t$, then $\rho(t + dt) - \rho(t) \approx a(t)dt$, where $dt$ denotes an infinitesimal time interval. If a jump occurred at time $t$ then $\rho(t + dt) - \rho(t) \approx a(t)dt + b(t)$. Since the jump locations are random, $\rho(t)$ is a stochastic process, hence the term SDE.

Now, this derivation can be generalized to the case where there are $M$ observation processes $N_t^{(1)}, N_t^{(2)}, \ldots, N_t^{(M)}$ with different rate functions $\lambda_1(X_t), \lambda_2(X_t), \ldots, \lambda_M(X_t)$. In this case the differential equations for the non-normalized posterior probabilities is

$$d\rho_i(t) = \sum_{j=1}^N Q_{ji}\rho_j(t)dt + \sum_{k=1}^M (\lambda_k(s_i) - 1)\rho_i(t)(dN_t^{(k)} - dt) \tag{15}$$

Recalling that $N_t^{(k)}$ is a counting process, namely $dN_t^{(k)}/dt = \sum_n \delta(t - t_n^k)$, we obtain (2), where $t_n^k$ is the arrival time of the $n$-th event in the $k$-th observation process.

# 5 Discussion

In this work we have introduced a linear recurrent neural network model capable of exactly implementing Bayesian state estimation and prediction from input spike trains in real time. The framework is mathematically rigorous and requires few assumptions, is naturally formulated in continuous time, and is based directly on spike train inputs, thereby sacrificing no temporal resolution. The setup is ideally suited to the integration of several sensory modalities, and retains its optimality in this setting as well. The linearity of the system renders an analytic solution possible, and a full characterization in terms of a-priori knowledge and online sensory input. This framework sets the stage for many possible extensions and applications, of which we mention several examples. (i) It is important to find a natural mapping between the current abstract neural model and more standard biological neural network models. One possible approach was mentioned in Section 2, but other options are possible and should be pursued. Additionally, the implementation of the estimation network (namely, the variables $\rho_i(t)$) using realistic spiking neurons is still open. (ii) At this point the matrix $Q$ in (4) is assumed to be known. Combining approaches to learning $Q$ and adapting the tuning curves $\lambda_k$ in real time will lend further plausibility and robustness to the system. (iii) The present framework, based on doubly stochastic Poisson processes, can be extended to more general point processes, using the filtering framework described in [10]. (iv) Currently, each world state is represented by a single neuron (a grandmother cell). This is clearly a non-robust representation, and it would be worthwhile to develop more distributed and robust representations. Finally, the problem of experimental verification of the framework is a crucial step in future work.

**Acknowledgments** The authors are grateful to Rami Atar his helpful advice on nonlinear filtering.

## Footnotes

[1]Implying that the rate function itself is a random process.

# References

[1] J.M. Beck and A. Pouget. Exact inferences in a neural implementation of a hidden markov model. *Neural Comput*, 19(5):1344–1361, 2007.

[2] R. Ben-Yishai, R.L. Bar-Or, and H. Sompolinsky. Theory of orientation tuning in visual cortex. *Proc Natl Acad Sci U S A*, 92(9):3844–8, Apr 1995. 542.

[3] P. Brémaud. *Point Processes and Queues: Martingale Dynamics*. Springer, New York, 1981.

[4] U.T. Eden, L.M. Frank, V. Solo, and E.N. Brown. Dynamic analysis of neural encoding by point process adaptive filtering. *Neural Computation*, 16:971–998, 2004.

[5] G.R. Grimmett and D.R. Stirzaker. *Probability and Random Processes*. Oxford University Press, third edition, 2001.

[6] A.H. Jazwinsky. *Stochastic Processes and Filtering Theory*. Academic Press, 1970.

[7] H.J. Kushner. Dynamical equations for optimal nonlinear filtering. *J. Differential Equations*, 3:179–190, 1967.

[8] R.P.N. Rao. Bayesian computation in recurrent neural circuits. *Neural Comput*, 16(1):1–38, 2004. 825.

[9] R.P.N. Rao. Neural models of Bayesain belief propagation. In K. Doya, S. Ishii, A. Pouget, and R. P. N. Rao, editors, *Bayesian Brain*, chapter 11. MIT Press, 2006.

[10] A. Segall, M. Davis, and T. Kailath. Nonlinear filtering with counting observations. *IEEE Tran. Information Theory,*, 21(2):143–149, 1975.

[11] S. Shoham, L.M. Paninski, M.R. Fellows, N.G. Hatsopoulos, J.P. Donoghue, and R.A. Norman. Statistical encoding model for a primary motor cortical brain-machine interface. *IEEE Trans Biomed Eng.*, 52(7):1312–22, 2005.

[12] D. L. Snyder. Filtering and detection for doubly stochastic Poisson processes. *IEEE Transactions on Information Theory*, IT-18:91–102, 1972.

[13] N. Twum-Danso and R. Brockett. Trajectory estimation from place cell data. *Neural Netw*, 14(6-7):835–844, 2001.

[14] W.M. Wonham. Some applications of stochastic differential equations to optimal nonlinear filtering. *J. SIAM Control*, 2(3):347–369, 1965.

[15] M. Zakai. On the optimal filtering of diffusion processes. *Z. Wahrscheinlichkeitheorie verw Gebiete*, 11:230–243, 1969.

